# ANN Based Classification for Heart Defibrillators

**M. Jabri, S. Pickard, P. Leong, Z. Chi, B. Flower, and Y. Xie**

Sydney University Electrical Engineering

NSW 2006 Australia

## Abstract

Current Intra-Cardia defibrillators make use of simple classification algorithms to determine patient conditions and subsequently to enable proper therapy. The simplicity is primarily due to the constraints on power dissipation and area available for implementation. Sub-threshold implementation of artificial neural networks offer potential classifiers with higher performance than commercially available defibrillators. In this paper we explore several classifier architectures and discuss micro-electronic implementation issues.

## 1.0 INTRODUCTION

Intra-Cardia Defibrillators (ICDs) represent an important therapy for people with heart disease. These devices are implanted and perform three types of actions:

    1.monitor the heart
    2.to pace the heart
    3.to apply high energy/high voltage electric shock

They sense the electrical activity of the heart through leads attached to the heart tissue. Two types of sensing are commonly used:

*Single Chamber:* Lead attached to the Right Ventricular Apex (RVA)
*Dual Chamber:* An additional lead is attached to the High Right Atrium (HRA).

The actions performed by defibrillators are based on the outcome of a classification procedure based on the heart rhythms of different heart diseases (abnormal rhythms or "arrhythmias").

There are tens of different arrhythmias of interest to cardiologists. They are clustered into three groups according to the three therapies (actions) that ICDs perform.

Figure 1 shows an example of a Normal Sinus Rhythm. Note the regularity in the beats. Of interest to us is what is called the QRS complex which represents the electrical activity in the ventricle during a beat. The R point represents the peak, and the distance between two heart beats is usually referred to as the RR interval.

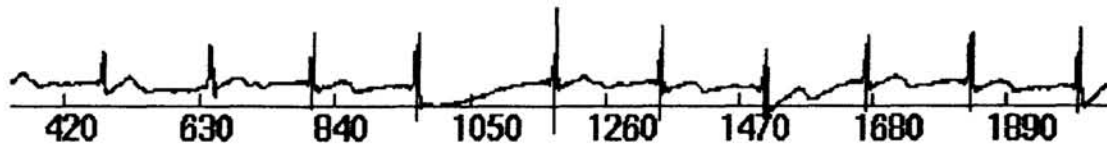

**FIGURE 1. A Normal Sinus Rhythm (NSR) waveform**

Figure 2 shows an example of a Ventricular Tachycardia (more precisely a Ventricular Tachycardia Slow or VTS). Note that the beats are faster in comparison with an NSR.

Ventricular Fibrillation (VF) is shown in Figure 3. Note the chaotic behavior and the absence of well defined heart beats.

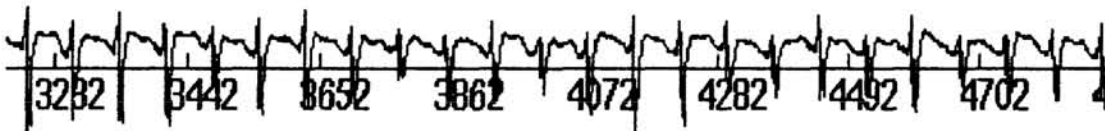

**FIGURE 2. A Ventricular Tachycardia (VT) waveform**

The three waveforms discussed above are examples of Intra-Cardia Electro-Grams (ICEG). NSR, VT and VF are representative of the type of action a defibrillator has to takes. For an NSR, an action of "continue monitoring" is used. For a VT an action of "pacing" is performed, whereas for VF a high energy/high voltage shock is issued. Because they are near-field signals, ICEGs are different from external Eltro-Cardio-Grams (ECG). As a result, classification algorithms developed for ECG patterns may not be necessarily valuable for ICEG recognition.

The difficulties in ICEG classification lie in that many arrhythmias share similar features and fuzzy situations often need to be dealt with. For instance, many ICDs make use of the heart rate as a fundamental feature in the arrhythmia classification process. But several arrhythmias that require different type of therapeutic actions have similar heart rates. For example, a Sinus Tachycardia (ST) is an arrhythmia characterized with a heart rate that is higher than that of an NSR and in the vicinity of a VT. Many classifier would classify an ST as VT leading to a therapy of pacing, whereas an ST is supposed to be grouped under an NSR type of therapy. Another example is a fast VT which may be associated with heart rates that are indicative of

VF. In this case the defibrillator would apply a VF type of therapy when only a VT type therapy is required (pacing).

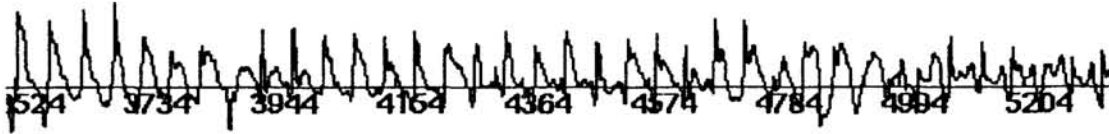

**FIGURE 3. A Ventricular Fibrillation (VF) waveform.**

The overlap of the classes when only heart rate is used as the main classification feature highlights the necessity of the consideration of further features with higher discrimination capabilities. Features that are commonly used in addition to the heart rate are:

1. average heart rate (over a period of time)
2. arrhythmia onset
3. arrhythmia probability density functions

Because of the limited power budget and area, arrhythmia classifiers in ICDs are kept extremely simple with respect to what could be achieved with more relaxed implementation constraints. As a result false positive (pacing or defibrillation when none is required) may be high and error rates may reach 13%.

Artificial neural network techniques offer the potential of higher classification performance. In order to maintain as lower power consumption as possible, VLSI micro-power implementation techniques need to be considered.

In this paper, we discuss several classifier architectures and sub-threshold implementation techniques. Both single and dual chamber based classifications are considered.

## 2.0 DATA

Data used in our experiments were collected from Electro-Physiological Studies (EPS) at hospitals in Australia and the UK. Altogether, and depending on whether single or dual chamber is considered, data from over 70 patients is available. Cardiologists from our commercial collaborator have labelled this data. All tests were performed on a testing set that was not used in classifier building. Arrhythmias recorded during EPS are produced by stimulation. As a result, no natural transitions are captured.

## 3.0 SINGLE CHAMBER CLASSIFICATION

We have evaluated several approaches for single chamber classification. It is important to note here that in the case of single chamber, not all arrhythmias could be correctly classified (not even by human experts). This is because data from the RVA lead represents mainly the ventricular electrical activity, and many atrial arrhythmias require atrial information for proper diagnosis.

## 3.1 MULTI-LAYER PERCEPTRONS

Table 1 shows the performance of multi-layer perceptrons (MLP) trained using vanilla back-propagation, conjugate gradient and a specialized limited precision training algorithm that we call Combined Search Algorithm (Xie and Jabri, 91). The input to the MLP are 21 features extracted from the time domain. There are three outputs representing three main groupings: NSR, VT and VF. We do not have the space here to elaborate on the choice of the input features. Interested readers are referenced to (Chi and Jabri, 91; Leong and Jabri, 91).

**TABLE 1. Performance of Multi-layer Perceptron Based Classifiers**

| Network | Training Algorithm | Precision | Average Performance |
|---------|--------------------|-----------|---------------------|
| 21-5-3  | backprop.          | unlimited | 96%                 |
| 21-5-3  | conj.-grad         | unlimited | 95.5%               |
| 21-5-3  | CSA                | 6 bits    | 94.8%               |

The summary here indicates that a high performance single chamber based classification can be achieved for ventricular arrhythmias. It also indicates that limited precision training does not significantly degrade this performance. In the case of limited precision MLP, 6 bits plus a sign bit are used to represent network activities and weights.

## 3.2 INDUCTION OF DECISION TREES

The same training data used to train the MLP was used to create a decision tree using the C4.5 program developed by Ross Quinlan (a derivative of the ID3 algorithm). The resultant tree was then tested, and the performance was 95% correct classification. In order to achieve this high performance, the whole training data had to be used in the induction process (windowing disabled). This has a negative side effect in that the trees generated tend to be large.

The implementation of decision trees in VLSI is not a difficult procedure. The problem how-ever is that because of the binary decision process, the branching thresholds are difficult to be implemented in digital (for large trees) and even more difficult to be implemented in micro-power analog. The latter implementation technique would be possible if the induction process can take advantage of the hardware characteristics in a similar way that "in-loop" training of sub-threshold VLSI MLP achieves the same objective.

## 4.0 DUAL CHAMBER BASED CLASSIFIERS

Two architectures for dual chamber based classifiers have been investigated: Multi-Module-Neural Networks and a hybrid Decision Tree/MLP. The difference between the classifier archi-tectures is a function of which arrhythmia group is being targetted for classification.

## 4.1 MULTI-MODULE NEURAL NETWORK

The multi-module neural network architecture aims at improving the performance with respect to the classification of Supra-Ventricular Tachycardia (SVT). The architecture is shown in Figure 4 with the classifier being the right most block.

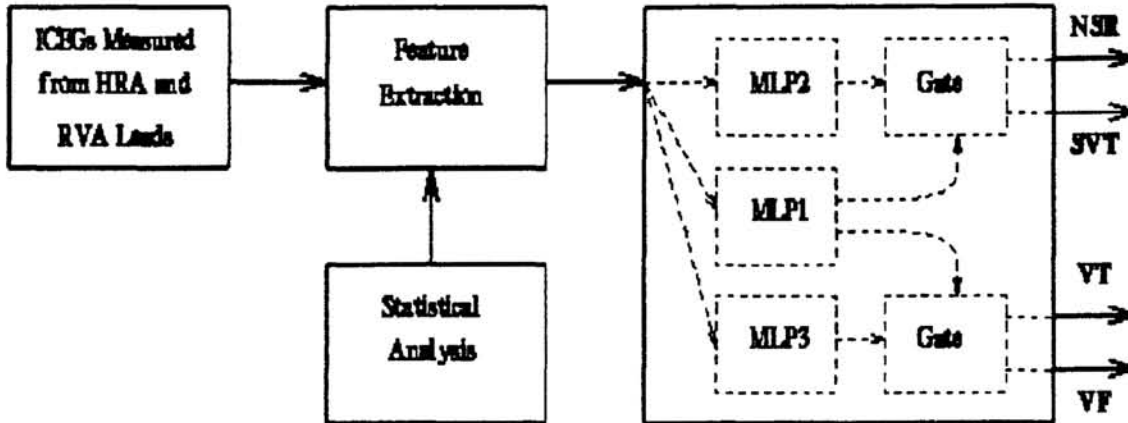

**FIGURE 4. Multi-Module Neural Network Classifier**

The idea behind this architecture is to divide the classification problem into that of discriminating between NSR and SVT on one hand and VF and VT on the other. The details of the operation and the training of this structured classifier can be found in (Chi and Jabri, 91).

In order to evaluate the performance of the MMNN classifier, a single large MLP was also developed. The single large MLP makes use of the same input features as the MMNN and targets the same classes. The performance comparison is shown in Table 2 which clearly shows that a higher performance is achieved using the MMNN.

## 4.2 HYBRID DECISION TREE/MLP

The hybrid decision tree/multi-layer perceptron "mimics" the classification process as performed by cardiologists. The architecture of the classifier is shown in Figure 5. The decision tree is used to produce a judgement on:

      1.The rate aspects of the ventricular and atrial channels,

      2.The relative timing between the atrial and ventricular beats.

In parallel with the decision tree, a morphology based classifier is used to perform template matching. The morphology classifier is a simple MLP with input that are signal samples (sampled at half the speed of the normal sampling rate of the signal).

The output of the timing and morphology classifiers are fed into an arbitrator which produces the class of the arrhythmia being observed. An "X out of Y" classifier is used to smooth out the

**TABLE 2. Performance of Multi-Module Neural Network Classifier and comparison with that of a single large MLP.**

| Rhythms | MMNN Best % | MMNN Worst % | Single MLP % |
|---------|-------------|--------------|--------------|
| NSR | 95.3 | 93.8 | 93.4 |
| ST | 98.6 | 98.6 | 97.5 |
| SVT | 96.4 | 93.3 | 95.4 |
| AT | 95.9 | 93.2 | 71.2 |
| AF | 86.7 | 85.4 | 77.5 |
| VT | 99.4 | 99.4 | 100 |
| VTF | 100 | 100 | 80.3 |
| VF | 97 | 97 | 99.4 |
| Average | 96.2 | 95.1 | 89.3 |
| SD | 4.18 | 4.8 | 11.31 |

classification output by the arbitrator and to produce an "averaged" final output class. Further details on the implementation and the operation of the hybrid classifier can be found in (Leong and Jabri, 91).

This classifier achieves a high performance classification over several types of arrhythmia. Table 3 shows the performance on a multi-patient database and indicate a performance of over 99% correct classification.

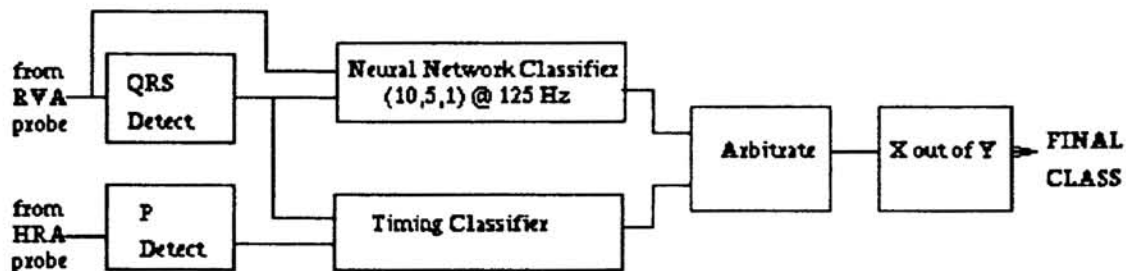

**FIGURE 5. Architecture of the hybrid decision tree/neural network classifier.**

## 5.0 MICROELECTRONIC IMPLEMENTATIONS

In all our classifier architecture investigations, micro-electronic implementation consider-ations were a constant constraint. Many other architectures that can achieve competitive per-formance were not discussed in this paper because of their unsuitability for low power/small area implementation. The main challenge in a low power/small area VLSI implementation of classifiers similar to those discussed above, is how to implement in very low power a MLP architecture that can reliably learn and achieve a performance comparable to that of the func-

tional simulations. Several design strategies can achieve the low power and small area objectives.

**TABLE 3. Performance of the hybrid decision tree/MLP classifier for dual chamber classification.**

| SubClass | Class | NSR | SVT | VT | VF |
|----------|-------|-----|-----|----|----|
| NSR | NSR | 5247 | 4 | 2 | 0 |
| ST | NSR | 1535 | 24 | 2 | 1 |
| SVT | SVT | 0 | 1022 | 0 | 0 |
| AT | SVT | 0 | 52 | 0 | 0 |
| AF | SVT | 0 | 165 | 0 | 0 |
| VT | VT | 0 | 0 | 322 | 0 |
| VT 1:1 | VT | 2 | 0 | 555 | 0 |
| VF | VF | 0 | 0 | 2 | 196 |
| VTF | VF | 0 | 2 | 0 | 116 |

Both digital and analog implementation techniques are being investigated and we report here on our analog implementation efforts only. Our analog implementations make use of the subthreshold operation mode of MOS transistors in order to maintain a very low power dissipation.

### 5.1 MASTER PERTURBATOR CHIP

The architecture of this chip is shown in Figure 6(a). Weights are implemented using a differential capacitor scheme refreshed from digital RAM. Synapses are implemented as four quadrant Gilbert multipliers (Pickard et al, 92). The chip has been fabricated and is currently being tested. The building blocks have so far been successfully tested. Two networks are implemented a 7-5-3 (total of 50 synapses) and a small single layer network. The single layer network has been successfully trained to perform simple logic operations using the Weight Perturbation algorithm (Jabri and Flower, 91).

### 5.2 THE BOURKE CHIP

The BOURKE chip (Leong and Jabri, 92) makes use of Multiplying Digital to Analog Converters to implement the synapses. Weights are stored in digital registers. All neurons were implemented as external resistors for the sake of evaluation. Figure 6(b) shows the schematics of a synapse. The BOURKE chip has a small network 3-3-1 and has been successfully tested (it was successfully trained to perform an XOR). A larger version of this chip with a 10-6-4 network is being fabricated.

## 6.0 Conclusions

We have presented in this paper several architectures for single and dual chamber arrhythmia

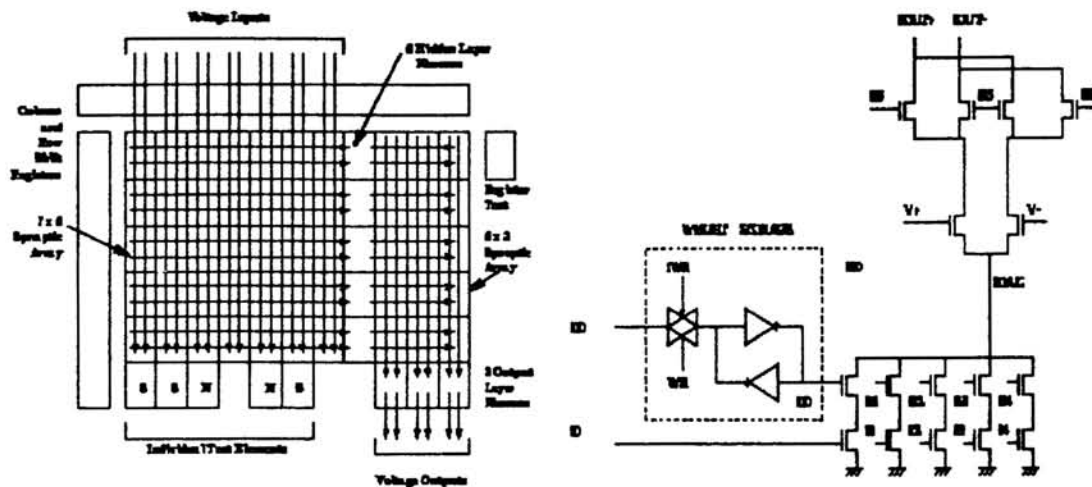

**FIGURE 6. (a) Architecture of the Master Perturbator chip. (b) Schematics of the BOURKE chip synapse implementation.**

classifiers. In both cases a good classification performance was achieved. In particular, for the case of dual chamber classification, the complexity of the problem calls on more structured classifier architectures. Two microelectronic low power implementation were briefly presented. Progress so far indicates that micro-power VLSI ANNs offer a technology that will enable the use of powerful classification strategies in implantable defibrillators.

## Acknowledgment

Work presented in this paper was supported by the Australian Department of Industry Technology & Commerce, Telectronics Pacing Systems, and the Australian Research Council.

## References

Z. Chi and M. Jabri (1991), "Identification of Supraventricular and Ventricular Arrhythmias Using a Combination of Three Neural Networks". Proceedings of the Computers in Cardiology Conference, Venice, Italy, September 1991.

M. Jabri and B. Flower (1991), ``Weight Perturbations: An optimal architecture and learning technique for analog VLSI feed-forward and recurrent multi-layer networks", Neural Computation, Vol. 3, No. 4, MIT Press.

P.H.W Leong and M. Jabri (1991), ``Arrhythmia Classification Using Two Intracardiac Leads", Proceedings of the Computers in Cardiology Conference, Venice, Italy.

P.H.W. Leong and M. Jabri (1992), ``An Analogue Low Power VLSI Neural Network", Proceedings of the Third Australian Conference on Neural Networks, pp. 147-150, Canberra, Australia.

S. Pickard, M. Jabri, P.H.W. Leong, B.G. Flower and P. Henderson (1992), ``Low Power Analogue VLSI Implementation of A Feed-Forward Neural Network", Proceedings of the Third Australian Conference on Neural Networks, pp. 88-91, Canberra, Australia.

Y. Xie and M. Jabri (1991), ``Training Algorithms for Limited Precision Feed-forward Neural Networks", submitted to IEEE Transactions on Neural Networks and Neural Computation.